# The Wisdom of Crowds in the Recollection of Order Information

**Mark Steyvers, Michael Lee, Brent Miller, Pernille Hemmer**
Department of Cognitive Sciences
University of California  Irvine
*mark.steyvers@uci.edu*

## Abstract

When individuals independently recollect events or retrieve facts from memory, how can we aggregate these retrieved memories to reconstruct the actual set of events or facts? In this research, we report the performance of individuals in a series of general knowledge tasks, where the goal is to reconstruct from memory the order of historic events, or the order of items along some physical dimension. We introduce two Bayesian models for aggregating order information based on a Thurstonian approach and Mallows model. Both models assume that each individual's reconstruction is based on either a random permutation of the unobserved ground truth, or by a pure guessing strategy. We apply MCMC to make inferences about the underlying truth and the strategies employed by individuals. The models demonstrate a "wisdom of crowds" effect, where the aggregated orderings are closer to the true ordering than the orderings of the best individual.

## 1      Introduction

Many demonstrations have shown that aggregating the judgments of a number of individuals results in an estimate that is close to the true answer, a phenomenon that has come to be known as the "wisdom of crowds" [1]. This was demonstrated by Galton, who showed that the estimated weight of an ox, when averaged across individuals, closely approximated the true weight [2]. Similarly, on the game show Who Wants to be a Millionaire, contestants are given the opportunity to ask all members of the audience to answer multiple choice questions. Over several seasons of the show, the modal response of the audience corresponded to the correct answer 91% of the time. More sophisticated aggregation approaches have been developed for multiple choice tasks, such as Cultural Consensus Theory, that additionally take differences across individuals and items into account [3]. The wisdom of crowds idea is currently used in several real-world applications, such as prediction markets [4], spam filtering, and the prediction of consumer preferences through collaborative filtering. Recently, it was shown that a form of the wisdom of crowds phenomenon also occurs within a single person [5]. Averaging multiple guesses from one person provides better estimates than the individual guesses.

We are interested in applying this wisdom of crowds phenomenon to human memory involving situations where individuals have to retrieve information more complex than single numerical estimates or answers to multiple choice questions. We will focus here on memory for order information. For example, we test individuals on their ability to reconstruct from memory the order of historic events (e.g., the order of US presidents), or the magnitude along some physical dimension (e.g., the order of largest US cities). We then develop computational models that infer distributions over orderings to explain the observed orderings across individuals. The goal is to demonstrate a wisdom of crowds effects where

the inferred orderings are closer to the actual ordering than the orderings produced by the majority of individuals.

Aggregating rank order data is not a new problem. In social choice theory, a number of systems have been developed for aggregating rank order preferences for groups (Marden, 1995). Preferential voting systems, where voters explicitly rank order their candidate preferences, are designed to pick one or several candidates out of a field of many. These systems, such as the Borda count, perform well in aggregating the individuals' rank order data, but with an inherent bias towards determining the top members of the list. However, as voting is a means for expressing individual preferences, there is no ground truth. The goal for these systems is to determine an aggregate of preferences that is in some sense "fair" to all members of the group. The rank aggregation problem has also been studied in machine learning and information retrieval [6,7]. For example, if one is presented with a ranked list of webpages from several search engines, how can these be combined to create a single ranking that is more accurate and less sensitive to spam?

Relatively little research has been done on the rank order aggregation problem with the goal of approximating a known ground truth. In follow-ups to Galton's work, some experiments were performed testing the ability of individuals to rank-order magnitudes in psychophysical experiments [8]. Also, an informal aggregation model for rank order data was developed for the Cultural Consensus Theory, using factor analysis of the covariance structure of rank order judgments [3]. This was used to (partially) recover the order of causes of death in the US on the basis of the individual orderings.

We present empirical and theoretical research on the wisdom of crowds phenomenon for rank order aggregation. No communication between people is allowed for these tasks, and therefore the aggregation method operates on the data produced by independent decision-makers. Importantly, for all of the problems there is a known ground truth. We compare several heuristic computational approaches—based on voting theory and existing models of social choice—that analyze the individual judgments and provide a single answer as output, which can be compared to the ground truth. We refer to these synthesized answers as "group" answers because they capture the collective wisdom of the group, even though no communication between group members occurred. We also apply probabilistic models based on a Thurstonian approach and Mallows model. The Thurstonian model represents the group knowledge about items as distributions on an interval dimension [9]. Mallows model is a distance-based model that represents the group answer as a modal ordering of items, and assumes each individual to have orderings that are more or less close to the modal ordering [10]. Although Thurstonian and Mallows type of models have often been used to analyze preference rankings [11], they have not been applied, as far as we are aware, to ordering problems where there is a ground truth. We also present extensions of these models that allow for the possibility of different response strategies—some individuals might be purely guessing because they have no knowledge of the problem and others might have partial knowledge of the ground truth. We develop efficient MCMC algorithms to infer the latent group orderings and assignments of individuals to response strategies. The advantage of MCMC estimation procedure is that it gives a probability distribution over group orderings, and we can therefore assess the likelihood of any particular group ordering.

## 2    Experiment

### 2.1    Method

Participants were 78 undergraduate students at the University of California, Irvine. The experiment was composed of 17 questions involving general knowledge regarding: population statistics (4 questions), geography (3 questions), dates, such as release dates for movies and books (7 questions), U.S. Presidents, material hardness, the 10 Commandments, and the first 10 Amendments of the U.S. Constitution. An interactive interface was presented on a computer screen. Participants were instructed to order the presented items (e.g., "Order these books by their first release date, earliest to most recent"), and responded by dragging the individual items on the screen to the desired location in the ordering. The initial ordering of the 10 items within a question was randomized across all questions and all participants.

Table 1: Unique orderings for each individual for the states and presidents ordering problems

```
A A A A A A A A A A A A A A A A A A A A A A A A A A A A A A A A A A A A A A A A B B A A A B A A B A A B A A A A A D A B A B E A D C B E C I J J H
B B B B B B B B B B B B B B B B B B B B B B C E B B B B B B B B B B B B B B B C F A C B B G A B B A B B F B F F B A C A F H E I J H B E G G J
C C C C C C C D D C C C C C C C C D C D D B B C D D D E C C D D F F C C F B B C D C C C C C D F D E C F E B D E G E C C I G H G I A B I I
D D D E D D E E C C D D D D E F F C F E C C D C G F F F D E C H C D D H C F D E A H I B F H C C H I B J C C I F I G E H A C B G H H D G
E E F D E F D F E F E E E F F D F C E E D F D H C D H F F F D J I H H I D I D D E F F B H A D A D I J H G I H E
F F E F F E F D F E F G H E D E E E G E E F F D H C C H G F E E E I E D I E D H D E F D D F D C F D C B A E H J C D F B F A A J D F E F
G G G H G G G H G I H F I G G H H G D H F H G E E G H G I J E F I H F D I E H I E F F I F C E E I G C C D D B J F H D F F F E C D
H I H H I I I H G H H I G G I I G I G I H G I G H G F I I H I E G H H I F G E G I G E J E E H H H G I J D H J H I C E F D D C A B C
I H I G H H I I I G F J H H H I I I H G I J H I I H I I H I I E I G G G G G I G G G G J G H H H G E G G G G G H H H G I J I F I B G B E C E B C F B
J J J J J J J J J J J J J J J J J J J J J J J G J J J J J J J H J J J J J J J J J J J J J J J J J I J J J J J J I G J E G J G J J G I A J D A A
0 1 1 2 2 2 2 2 3 3 3 3 3 3 3 4 4 4 4 5 5 5 5 6 6 6 6 6 7 7 7 7 7 7 8 8 8 9 9 9 10 10 10 11 11 12 13 14 14 16 18 20 22 24 26 26 30 37 42
2 1 5 1 1 1 1 1 3 2 1 1 1 1 1 1 1 1 1 1 1 1 1 1 1 1 1 1 1 1 1 1 1 1 1 1 1 1 1 1 1 1 1 1 1 1 1 1 1 1 1 1 1 1 1 1 1 1 1
```

A = Oregon
B = Utah
C = Nebraska
D = Iowa
E = Alabama
F = Ohio
G = Virginia
H = Delaware
I = Connecticut
J = Maine

```
A A A A A A A A A A A A A A A A A A A A A A A A A A A A A A A A A A A A A A A A A A A A A A A A A A A A A A A A A A A A A C A H
B B B B C B B B C B B B B B B B C D B B B B C D B B B B C D B B B B B C D B B B B E C D B C D C D C C C C B D E C E C E C G J C
C C C C B C C C B C C C C B C C C C D D C C C C C C C C E B C C C C D B B B B E C D B B B E C B C D C D C C C C B D E C E C E C G J G D
D D D E D D E E D D D E E D D E C E C E D D E E C I D D D D F D E E E E D G E D G I G G J G F C B D D D D E D I E G D F C J C J E F B J I E G J J C E D J
E E E D E E D D D E E D D E C I D D D D F D E E E E D G E D G I G G J G F C B D D D D E D I E G D F C J C J E F B J I E G J J C E D J
F F G F F F F G F F F G F F H F G F F F E F F H H D I G H F J J D J F D D D H E H D G G H J F G I J H J H B H E G D D G I J I F D B I H J B C E
G G F G G H H F G G G F G J G H F I I G J J G G J G I F I I G I F I I G F G D I F H J H G J J G F H E I I D H J I C H G D J G C D I F I H G
H H H H H G G H H J I I H G F I J G G F H I J H G F J J H H H H G H J H H H H H H H G H J D H H G G B F H G H F F A
I J I I I J I I J J H J I J J H J H J I J J H J H J I G F F I H F G G I F G H J H H J I F G H I G F I E F I D G D G J H I E H D D G G B F H G H F F A
J I J J J J J J H H J I H H J I H I G I H J J H G J J H H H G J H H H H G H J D H I J J I J J D J F G F F E H F D I J F D B H E B
0 1 1 1 2 2 2 2 3 3 3 3 4 4 4 4 4 6 6 6 6 6 6 6 6 7 7 7 7 7 7 7 8 8 8 8 8 9 9 9 9 9 10 10 10 10 11 12 13 13 13 14 14 14 14 15 17 18 19 26 28
5 1 2 1 1 1 1 3 1 1 1 1 1 1 1 1 1 1 1 1 1 1 1 1 1 1 1 1 1 1 1 1 1 1 1 1 1 1 1 1 1 1 1 1 1 1 1 1 1 1 1 1 1 1 1 1 1 1 1 1 1
```

A = George Washington
B = John Adams
C = Thomas Jefferson
D = James Jackson
E = Andrew Jackson
F = Theodore Roosevelt
G = Woodrow Wilson
H = Franklin D. Roosevelt
I = Harry S. Truman
J = Dwight D. Eisenhower

## 2.2 Results

To evaluate the performance of participants as well as models, we measured the distance between the reconstructed and the correct ordering. A commonly used distance metric for orderings is Kendall's $\tau$. This distance metric counts the number of adjacent pairwise disagreements between orderings. Values of $\tau$ range from: $0 \leq \tau \leq N(N-1)/2$, where $N$ is the number of items in the order (10 for all of our questions). A value of zero means the ordering is exactly right, and a value of one means that the ordering is correct except for two neighboring items being transposed, and so on up to the maximum possible value of 45.

Table 1 shows all unique orderings, by column, that were produced for two problems: arranging U.S. States by east-west location, and sorting U.S. Presidents by the time they served in office. The correct ordering is shown on the right. The columns are sorted by Kendall's $\tau$ distance. The first and second number below each ordering correspond to Kendall's $\tau$ distance and the number of participants who produced the ordering respectively. These two examples show that only a small number of participants reproduced the correct ordering (in fact, for 11 out of 17 problems, no participant gave the correct answer). It also shows that very few orderings are produced by multiple participants. For 8 out of 17 problems, each participant produced a unique ordering.

To summarize the results across participants, the column labeled PC in Table 2 shows the proportion of individuals who got the ordering exactly right for each of the ordering task questions. On average, about one percent of participants recreated the correct rank ordering perfectly. The column $\tau$, shows the mean $\tau$ values over the population of participants for each of the 17 sorting task questions. As this is a prior knowledge task, it is interesting to note the best performance overall was achieved on the Presidents, States from west to east, Oscar movies, and Movie release dates tasks. These four questions relate to educational and cultural knowledge that seems most likely to be shared by our undergraduate subjects.

Finally, an important summary statistic is the performance of the best individual. Instead of picking the best individual separately for each problem, we find the individual who scores best across all problems. Table 2, bottom row, shows that this individual has on average a $\tau$ distance of 7.8. To demonstrate the wisdom of crowds effect, we have to show that the synthesized group ordering outperforms the ordering, on average, of this best individual.

## 3 Modeling

We evaluated a number of aggregation models on their ability to reconstruct the ground truth based on the group ordering inferred from individual orderings. First, we evaluate two heuristic methods from social choice theory based on the mode and Borda counts. One drawback of such heuristic aggregation models is that they create no explicit representation of each individual's working knowledge. Therefore, even though such methods can aggregate

Table 2: Performance of the four models and human participants

| Problem | Humans | | Thurstonian Model | | | Mallows Model | | | Borda Counts | | | Mode | | |
|---|---|---|---|---|---|---|---|---|---|---|---|---|---|---|
| | PC | τ | C | τ | Rank | C | τ | Rank | C | τ | Rank | C | τ | Rank |
| books | .000 | 12.3 | 0 | 5 | 91 | 0 | 5 | 91 | 0 | 7 | 82 | 0 | 12 | 40 |
| city population europe | .000 | 16.9 | 0 | 11 | 81 | 0 | 12 | 77 | 0 | 11 | 81 | 0 | 17 | 42 |
| city population us | .000 | 15.9 | 0 | 7 | 96 | 0 | 7 | 96 | 0 | 12 | 67 | 0 | 16 | 45 |
| city population world | .000 | 19.3 | 0 | 16 | 73 | 0 | 16 | 73 | 0 | 15 | 77 | 0 | 19 | 44 |
| country landmass | .000 | 10.9 | 0 | 5 | 95 | 0 | 5 | 95 | 0 | 5 | 95 | 0 | 7 | 76 |
| country population | .000 | 14.6 | 0 | 12 | 74 | 0 | 11 | 82 | 0 | 11 | 82 | 0 | 15 | 53 |
| hardness | .000 | 15.3 | 0 | 14 | 64 | 0 | 14 | 64 | 0 | 11 | 91 | 0 | 15 | 46 |
| holidays | .051 | 8.9 | 0 | 4 | 78 | 0 | 5 | 77 | 0 | 4 | 78 | 1 | 0 | 100 |
| movies releasedate | .013 | 7.3 | 0 | 2 | 95 | 0 | 2 | 95 | 0 | 2 | 95 | 0 | 2 | 95 |
| oscar bestmovies | .013 | 11.2 | 0 | 4 | 90 | 0 | 4 | 90 | 0 | 3 | 97 | 0 | 3 | 97 |
| oscar movies | .000 | 11.9 | 0 | 1 | 100 | 0 | 1 | 100 | 0 | 2 | 96 | 0 | 2 | 96 |
| presidents | .064 | 7.5 | 0 | 2 | 87 | 0 | 1 | 94 | 0 | 3 | 79 | 1 | 0 | 100 |
| rivers | .000 | 16.1 | 0 | 13 | 77 | 0 | 14 | 67 | 0 | 11 | 91 | 0 | 16 | 42 |
| states westeast | .026 | 8.2 | 0 | 2 | 88 | 0 | 2 | 88 | 0 | 3 | 78 | 0 | 1 | 97 |
| superbowl | .000 | 18.6 | 0 | 16 | 65 | 0 | 15 | 71 | 0 | 10 | 96 | 0 | 19 | 40 |
| ten amendments | .013 | 14.0 | 0 | 2 | 97 | 0 | 3 | 96 | 0 | 5 | 90 | 0 | 4 | 95 |
| ten commandments | .000 | 16.8 | 0 | 8 | 90 | 0 | 7 | 91 | 0 | 12 | 74 | 0 | 17 | 51 |
| AVERAGE | .011 | 13.3 | .00 | 7.29 | 84.8 | .00 | 7.29 | 85.1 | .00 | 7.47 | 85.3 | .12 | 9.67 | 68.2 |
| BEST INDIVIDUAL | 0 | 7.8 | | | | | | | | | | | | |

the individual pieces of knowledge across individuals, they cannot explain why individuals rank the items in a particular way. To address this potential weakness, we develop two simple probabilistic models based on the Thurstonian approach [9] and Mallows model [10].

## 3.1 Heuristic Models

We tested two heuristic aggregation models. In the simplest heuristic, based on the mode, the group answer is based on the most frequently occurring sequence of all observed sequences. In cases where several different sequences correspond to the mode, a randomly chosen modal sequence was picked. The second method uses the Borda count method, a widely used technique from voting theory. In the Borda count method, weighted counts are assigned such that the first choice "candidate" receives a count of $N$ (where $N$ is the number of candidates), the second choice candidate receives a count of $N$-1, and so on. These counts are summed across candidates and the candidate with the highest count is considered the "most preferred". Here, we use the Borda count to create an ordering over all items by ordering the Borda counts.

Table 2 reports the performance of all of the aggregation models. For each, we checked whether the inferred group order is correct ($C$) and measured Kendall's $\tau$. We also report in the rank column the percentage of participants who perform worse or the same as the group answer, as measured by $\tau$. With the rank statistic, we can verify the wisdom of crowds effect. In an ideal model, the aggregate answer should be as good as or better than all of the individuals in the group. Table 1 shows the results separately for each problem, and averaged across all the problems.

These results show that the mode heuristic leads to the worst performance overall in rank. On average, the mode is as good or better of an estimate than 68% of participants. This means that 32% of participants came up with better solutions individually. This is not surprising, since, with an ordering of 10 items, it is possible that only a few participants will agree on the ordering of items. The difficulty in inferring the mode makes it an unreliable method for constructing a group answer. This problem will be exacerbated for orderings involving more than 10 items, as the number of possible orderings grows combinatorially. The Borda count method performs relatively well in terms of Kendall's $\tau$ and overall rank performance. On average, these methods perform with ranks of 85%, indicating that the group answers from these methods score amongst the best individuals. On average, the Borda count has an average distance of 7.47, which outperforms the best individual over all problems.

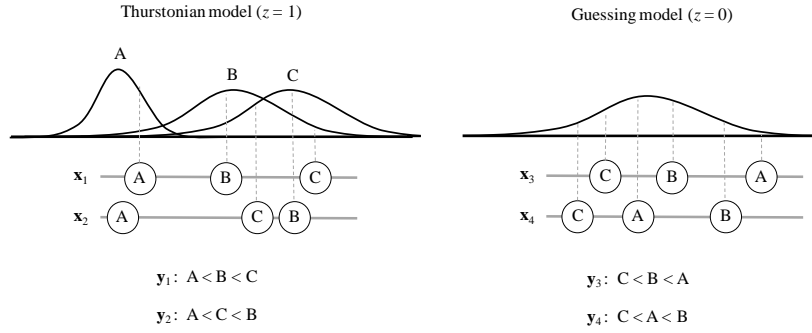

Figure 1. Illustration of the extended Thurstonian Model with a guessing component

## 3.2    A Thurstonian Model

In the Thurstonian approach, the overall item knowledge for the group is represented explicitly as a set of coordinates on an interval dimension. The interval representation is justifiable, at least for some of the problems in our study that involve one-dimensional concepts, such as the relative timing of events, or the lengths of items. We will introduce an extension of the Thurstonian approach where the orderings of some of the individuals are drawn from a Thurstonian model and others are drawn are based on a guessing process with no relation to the underlying interval representation.

To introduce the basic Thurstonian approach, let $N$ be the number of items in the ordering task and $M$ the number of individuals ordering the items. Each item is represented as a value $\mu_i$ along this dimension, where $i \in \{1, ..., N\}$. Each individual is assumed to have access to this group-level information. However, individuals might not have precise knowledge about the exact location of each item. We model each individual's location of the item by a single sample from a Normal distribution, centered on the item's group location. Specifically, in our model, when determining the order of items, a person $j \in \{1, ..., M\}$ samples a value from each item $i$, $x_{ij} \sim \mathrm{N}(\mu_i, \sigma_i)$. The standard deviation $\sigma_i$ captures the uncertainty that individuals have about item $i$ and the samples $x_{ij}$ represent the mental representation for the individual. The ordering for each individual is then based on the ordering of their samples. Let $\mathbf{y}_j$ be the observed ordering of the items for individual $j$ so that $\mathbf{y}_j = (i_1, i_2, ..., i_N)$ if and only if $x_{i_1 j} < x_{i_2 j} < \cdots < x_{i_N j}$. Figure 1 (left panel) shows an example of this basic Thurstonian model with group-level information for three items, A, B, and C and two individuals. In the illustration, there is a larger degree of overlap between the representations for B and C making it likely that items B and C are transposed (as illustrated for the second individual).

We extend this basic Thurstonian model by incorporating a guessing component. We found this to be a necessary extension because some individuals in the ordering tasks actually were

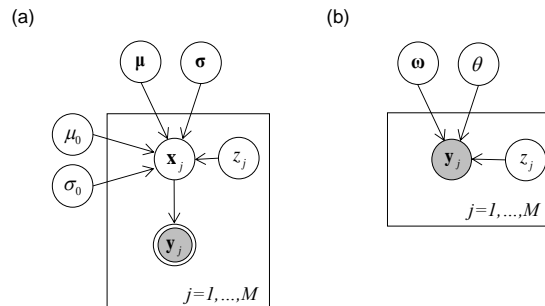

Figure 2. Graphical model of the extended Thurstonian Model (a) and Mallows model (b)

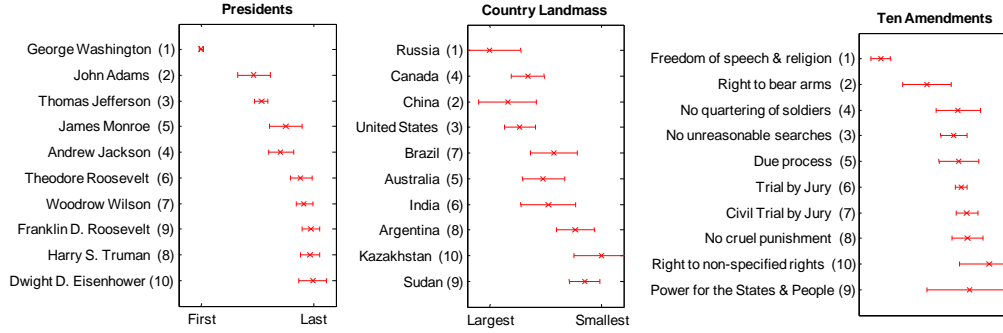

Figure 3. Sample Thurstonian inferred distributions. The vertical order is the ground truth ordering, while the numbers in parentheses show the inferred group ordering

not familiar with any of the items in the ordering tasks (such as the Ten Commandments or ten amendents). In the extended Thurstonian model, the ordering of such cases are assumed to originate from a single distribution, $x_{ij} \sim N(\mu_0, \sigma_0)$, where no distinction is made between the different items—all samples come from the same distribution with parameters $\mu_0, \sigma_0$. Therefore, the orderings produced by the individuals under this model are completely random. For example, Figure 1, right panel shows two orderings produced from this guessing model. We associate a latent state $z_j$ with each individual that determines whether the ordering from each individual is produced by the guessing model or the Thurstonian model:

$$x_{ij} \mid \mu_i, \sigma_i, \mu_0, \sigma_0 \sim \begin{cases} N(\mu_i, \sigma_i) & z_j = 1 \\ N(\mu_0, \sigma_0) & z_j = 0. \end{cases} \qquad (1)$$

To complete the model, we placed a standard prior on all normal distributions, $p(\mu, \sigma) \propto 1/\sigma^2$. Figure 2a shows the graphical model for the Thurstonian model. Although the model looks straightforward as a hierarchical model, inference in this model has proven to be difficult because the observed variable $y_j$ is a deterministic ranking function (indicated by the double bordered circle) of the underlying latent variable $x_j$. The basic Thurstonian model was introduced by Thurstone in 1927, but only recently have MCMC methods been developed for estimation [12]. We developed a simplified MCMC procedure as described in the supplementary materials that allows for efficient estimation of the underlying true ordering, as well as the assignment of individuals to response strategies.

The results of the extended Thurstonian model are shown in Table 2. The model performs approximately as well as the Borda count method. The model does not recover the exact answer for any of the 17 problems, based on the knowledge provided by the 78 participants. It is possible that a larger sample size is needed in order to achieve perfect reconstructions of the ground truth. However, the model, on average, has an distance of 7.29 from the actual truth, which is better than the best individual over all problems.

One advantage of the probabilistic approach is that it gives insight into the difficulty of the task and the response strategies of individuals. For some problems, such as the Ten Commandments, 32% of individuals were assigned to the guessing strategy ($z_j = 0$). For other problems, such as the US Presidents, only 16% of individuals were assigned to the guessing strategy, indicating that knowledge about this domain was more widely distributed in our group of individuals. Therefore, the extension of the Thurstonian model can eliminate individuals who are purely guessing the answers.

An advantage of the representation underlying the Thurstonian model is that it allows a visualization of group knowledge not only in terms of the order of items, but also in terms of the uncertainty associated with each item on the interval scale. Figure 3 shows the inferred distributions for four problems where the model performed relatively well. The crosses correspond to the mean of $\mu_i$ across all samples, and the error bars represent the standard

deviations $\sigma_i$ based on a geometric average across all samples. These visualizations are intuitive, and show how some items are confused with others in the group population. For instance, nearly all participants were able to identify Maine as the easternmost state in our list, but many confused the central states. Likewise, there was a large agreement on the proper placement of "the right to bear arms" in the amendments question — this amendment is often popularly referred to as "The Second Amendment".

### 3.3   Mallows Model

One drawback of the Thurstonian model is that it gives an analog representation for each item, which might be inappropriate for some problems. For example, it seems psychologically implausible that the ten amendments or Ten Commandments are mentally represented as coordinates on an interval scale. Therefore, we also applied probabilistic models where the group answer is based on a pure rank ordering. One such a model is Mallows model [7, 9, 10], a distance-based model that assumes that observed orderings that are close to the group ordering are more likely than those far away. One instantiation of Mallows model is based on Kendall's distance to measure the number of pairwise permutations between the group order and the individual order. Specifically, the probability of any observed order $y$, given the group order $\omega$ is:

$$p(y|\omega,\theta) = \frac{1}{\Psi(\theta)} e^{-\theta d(y,\omega)} \tag{2}$$

where $d$ is the Kendall's distance. The scaling parameter $\theta$ determines how close the observed orders are to the group ordering. As described by [7], the normalization function $\Psi(\theta)$ does not depend on $\omega$ and can be calculated efficiently by:

$$\Psi(\theta) = \prod_{i=1}^{N} \frac{1-e^{-i\theta}}{1-e^{-\theta}} . \tag{3}$$

The model as stated in the Eqs. (2) and (3) describe that standard Mallows model that has often been used to model preference ranking data. We now introduce a simple variant of this model that allows for contaminants. The idea is that some of the individuals orderings do not originate at all from some common group knowledge, and instead are based on a guessing process. The extended model introduces a latent state $z_j$ where $z_j = 1$ if the individual $j$ produced the ordering based on Mallows model and $z_j = 0$ if the individual is guessing. We model guessing by choosing an ordering uniformly from all possible orderings of $N$ items. Therefore, in the extended model, we have

$$p(y_j|\omega,\theta,z_j) = \begin{cases} \frac{1}{\Psi(\theta)} e^{-\theta d(y,\omega)} & z_j = 1 \\ 1/N! & z_j = 0. \end{cases} \tag{4}$$

To complete the model, we place a Bernoulli(1/2) prior over $z_j$. The MCMC inference algorithm to estimate the distribution over $\omega$, $z$ and $\theta$ given the observed data is based on earlier work   [6]. We extended the algorithm to estimate $z$ and also allow for the efficient estimation of $\theta$. The details of the inference procedure are described in the supplementary materials.

The result of the inference algorithm is a probability distribution over group answers $\omega$, of which we take the mode as the single answer for a particular problem. Note that the inferred group ordering does not have to correspond with an ordering of any particular individual. The model just finds the ordering that is close to all of the observed orderings, except those that can be better explained by a guessing process. Figure 4 illustrates the model solution based on a single MCMC sample for the Ten Commandments and ten amendment sorting tasks. The figure shows the distribution of distances from the inferred group ordering. Each circle corresponds to an individual. Individuals assigned to Mallows model and the guessing model are illustrated by filled and unfilled circles respectively. The solid and dashed red lines show the expected distributions based on the model parameters. Note that although Mallows model describes an exponential falloff in probability based on the distance from the group ordering, the expected distributions also take into account the number of orderings that exist at each distance (see [11], page 79, for a recursive algorithm to compute this).

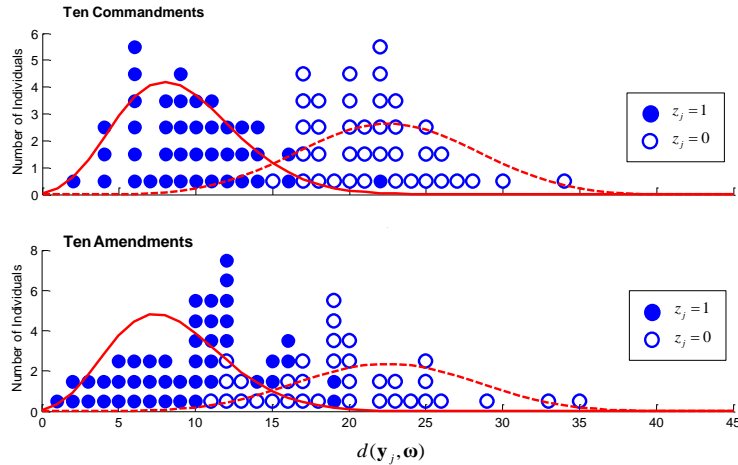

Figure 4. Distribution of distances from group answer for two example problems.

Figure 4 shows the distribution over individuals that are captured by the two routes in the model. The individuals with a Kendall's $\tau$ above below 15 tend to be assigned to Mallows route and all other individuals are assigned to the the guessing route. Interestingly, the distribution over distances appears to be bimodal, especially for the Ten Commandments. The middle peak of the distribution occurs at 22, which is close to the expected value of 22.5 based on guessing. This result seems intuitively plausible -- not everybody has studied the Ten Commandments, let alone the order in which they occur.

Table 2 shows the results for the extended Mallows model across all 17 problems. The overall performance, in terms of Kendall's $\tau$ and rank is comparable to the Thurstonian model and the Borda count method. Therefore, there does not appear to be any overall advantage of this particular approach. For the Ten Commandments and ten amendment sorting tasks, Mallows model performs the same or better than the Thurstonian model. This suggests that for particular ordering tasks, where there is arguably no underlying analog representation, a pure rank-ordering representation such as Mallows model might have an advantage.

## 4    Conclusions

We have presented two heuristic aggregation approaches, as well as two probabilistic approaches, for the problem of aggregating rank orders to uncover a ground truth. For each problem, we found that there were individuals who performed better than the aggregation models (although we cannot identify these individuals until after the fact). However, across all problems, no person consistently outperformed the model. Therefore, for all aggregation methods, except for the mode, we demonstrated a wisdom of crowds effect, where the average performance of the model was better than the best individual over all problems.

We also presented two probabilistic approaches based on the classic Thurstonian and Mallows approach. While neither of these models outperformed the simple Borda count heuristic models, they do have some advantages over them. The Thurstonian model not only extracts a group ordering, but also a representation of the uncertainty associated with the ordering. This can be visualized to gain insight into mental representations and processes. In addition, the Thurstonian and Mallows models were both extended with a guessing component to allow for the possibility that some individuals simply do not know any of the answers for a particular problem. Finally, although not explored here, the Bayesian approach potentially offers advantages over heuristic approaches because the probabilistic model can be easily expanded with additional sources of knowledge, such as confidence judgments from participants and background knowledge about the items.

## References

[1] Surowiecki, J. (2004). The Wisdom of Crowds. New York, NY: W. W. Norton & Company, Inc.

[2] Galton, F. (1907). Vox Populi. Nature, 75, 450-451.

[3] Romney, K. A., Batchelder, W. H., Weller, S. C. (1987). Recent Applications of Cultural Consensus Theory. American Behavioral Scientist, 31, 163-177.

[4] Dani, V., Madani, O., Pennock, D.M., Sanghai, S.K., & Galebach, B. (2006). An Empirical Comparison of Algorithms for Aggregating Expert Predictions. In Proceedings of the Conference on Uncertainty in Artificial Intelligence (UAI).

[5] Vul, E & Pashler, H (2008). Measuring the Crowd Within: Probabilistic representations Within individuals. *Psychological Science, 19*(7) 645-647.

[6] Lebanon, G. & Lafferty, J. (2002). Cranking: Combining Rankings using Conditional Models on Permutations. Proc. of the 19th International Conference on Machine Learning.

[7] Lebanon, G., & Mao, Y. (2008). Non-Parametric Modeling of Partially Ranked Data. Journal of Machine Learning Research, 9, 2401-2429.

[8] Gordon, K. (1924). Group Judgments in the Field of Lifted Weights. Journal of Experimental Psychology, 7, 398-400.

[9] Thurstone, L. L. (1927). A law of comparative judgement. Psychological Review, 34, 273–286.

[10] Mallows, C.L. (1957). Non-null ranking models, Biometrika, 44:114–130.

[11] Marden, J. I. (1995). Analyzing and Modeling Rank Data. New York, NY: Chapman & Hall USA.

[12] Yao, G., & Böckenholt, U. (1999). Bayesian estimation of Thurstonian ranking models based on the Gibbs sampler. British Journal of Mathematical and Statistical Psychology, 52, 79–92.
